# Manifold Parzen Windows

**Pascal Vincent and Yoshua Bengio**
Dept. IRO, Université de Montréal
C.P. 6128, Montreal, Qc, H3C 3J7, Canada
*{vincentp,bengioy}@iro.umontreal.ca*
http://www.iro.umontreal.ca/∼vincentp

## Abstract

The similarity between objects is a fundamental element of many learning algorithms. Most non-parametric methods take this similarity to be fixed, but much recent work has shown the advantages of learning it, in particular to exploit the local invariances in the data or to capture the possibly non-linear manifold on which most of the data lies. We propose a new non-parametric kernel density estimation method which captures the local structure of an underlying manifold through the leading eigenvectors of regularized local covariance matrices. Experiments in density estimation show significant improvements with respect to Parzen density estimators. The density estimators can also be used within Bayes classifiers, yielding classification rates similar to SVMs and much superior to the Parzen classifier.

## 1 Introduction

In [1], while attempting to better understand and bridge the gap between the good performance of the popular Support Vector Machines and the more traditional K-NN (K Nearest Neighbors) for classification problems, we had suggested a modified Nearest-Neighbor algorithm. This algorithm, which was able to slightly outperform SVMs on several real-world problems, was based on the geometric intuition that the classes actually lived "close to" a lower dimensional non-linear manifold in the high dimensional input space. When this was not properly taken into account, as with traditional K-NN, the sparsity of the data points due to having a finite number of training samples would cause "holes" or "zig-zag" artifacts in the resulting decision surface, as illustrated in Figure 1.

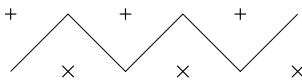

**Figure 1:** *A local view of the decision surface, with "holes", produced by the Nearest Neighbor when the data have a local structure (horizontal direction).*

The present work is based on the same underlying geometric intuition, but applied to the well known Parzen windows [2] non-parametric method for density estimation, using Gaussian kernels.

Most of the time, Parzen Windows estimates are built using a "spherical Gaussian" with a single scalar variance (or *width*) parameter $\sigma^2$. It is also possible to use a "diagonal Gaussian", i.e. with a diagonal covariance matrix, or even a "full Gaussian" with a full covariance matrix, usually set to be proportional to the global empirical covariance of the

training data. However these are equivalent to using a spherical Gaussian on preprocessed, *normalized* data (i.e. normalized by subtracting the empirical sample mean, and multiplying by the inverse sample covariance). Whatever the shape of the kernel, if, as is customary, a *fixed shape* is used, merely centered on every training point, the shape can only compensate for the *global structure* (such as global covariance) of the data.

Now if the true density that we want to model is indeed "close to" a non-linear lower dimensional manifold embedded in the higher dimensional input space, in the sense that most of the probability density is concentrated around such a manifold (with a small noise component away from it), then using Parzen Windows with a spherical or fixed-shape Gaussian is probably not the most appropriate method, for the following reason.

While the true density mass, in the vicinity of a particular training point $x_i$, will be mostly concentrated in a few local directions along the manifold, a spherical Gaussian centered on that point will spread its density mass equally along all input space directions, thus giving too much probability to irrelevant regions of space and too little along the manifold. This is likely to result in an excessive "bumpyness" of the thus modeled density, much like the "holes" and "zig-zag" artifacts observed in KNN (see Fig. 1 and Fig. 2).

If the true density in the vicinity of $x_i$ is concentrated along a lower dimensional manifold, then it should be possible to infer the local direction of that manifold from the neighborhood of $x_i$, and then anchor on $x_i$ a Gaussian "pancake" parameterized in such a way that it spreads mostly along the directions of the manifold, and is almost flat along the other directions. The resulting model is a mixture of Gaussian "pancakes", similar to [3], mixtures of probabilistic PCAs [4] or mixtures of factor analyzers [5, 6], in the same way that the most traditional Parzen Windows is a mixture of spherical Gaussians. But it remains a memory-based method, with a Gaussian kernel centered on each training points, yet with a *differently shaped* kernel for each point.

## 2    The Manifold Parzen Windows algorithm

In the following we formally define and justify in detail the proposed algorithm. Let $\mathcal{X}$ be an $n$-dimensional random variable with values in $\mathbb{R}^n$, and an *unknown probability density function* $p_{\mathcal{X}}(.)$. Our *training set* contains $l$ samples of that random variable, collected in a $l \times n$ matrix $X$ whose row $x_i$ is the $i$-th sample. Our goal is to estimate the density $p_{\mathcal{X}}$. Our estimator $\hat{p}_{mp}(.)$ has the form of a mixture of Gaussians, but unlike the Parzen density estimator, its covariances $C_i$ are not necessarily spherical and not necessarily identical everywhere:

$$\hat{p}_{mp}(x) = \frac{1}{l} \sum_{i=1}^{l} \mathcal{N}_{x_i, C_i}(x),\tag{1}$$

where $\mathcal{N}_{\mu, C}(x)$ is the multivariate Gaussian density with mean vector $\mu$ and covariance matrix $C$:

$$\mathcal{N}_{\mu, C}(x) = \frac{1}{\sqrt{(2\pi)^n |C|}} e^{-\frac{1}{2}(x-\mu)'C^{-1}(x-\mu)}\tag{2}$$

where $|C|$ is the determinant of $C$. How should we select the individual covariances $C_i$? From the above discussion, we expect that if there is an underlying "non-linear principal manifold", those gaussians would be "pancakes" aligned with the plane locally tangent to this underlying manifold. The only available information (in the absence of further prior knowledge) about this tangent plane can be gathered from the training samples int the neighborhood of $x_i$. In other words, we are interested in computing the *principal directions* of the samples in the neighborhood of $x_i$.

For generality, we can define a *soft neighborhood* of $x_i$ with a neighborhood kernel $\mathcal{K}(x; x_i)$ that will associate an influence weight to any point $x$ in the neighborhood of

$x_i$. We can then compute the weighted covariance matrix

$$C_{\mathcal{K}_i} = \frac{\sum_{j=1..l, j \neq i} \mathcal{K}(x_j; x_i)(x_j - x_i)'(x_j - x_i)}{\sum_{j=1..l, j \neq i} \mathcal{K}(x_j; x_i)} \qquad (3)$$

where $(x_j - x_i)'(x_j - x_i)$ denotes the outer product.

$\mathcal{K}(x, x_i)$ could be a spherical Gaussian centered on $x_i$ for instance, or any other positive definite kernel, possibly incorporating prior knowledge as to what constitutes a reasonable neighborhood for point $x_i$. Notice that if $\mathcal{K}(x, x_i)$ is a constant (uniform kernel), $C_{\mathcal{K}_i}$ is the global training sample covariance. As an important special case, we can define a *hard k-neighborhood* for training sample $x_i$ by assigning a weight of 1 to any point no further than the $k$-th nearest neighbor of $x_i$ among the training set, according to some metric such as the Euclidean distance in input space, and assigning a weight of 0 to points further than the $k$-th neighbor. In that case, $C_{\mathcal{K}_i}$ is the unweighted covariance of the $k$ nearest neighbors of $x_i$.

Notice what is happening here: we start with a possibly rough prior notion of neighborhood, such as one based on the ordinary Euclidean distance in input space, and use this to compute a local covariance matrix, which implicitly defines a refined local notion of neighborhood, taking into account the local direction observed in the training samples.

Now that we have a way of computing a local covariance matrix for each training point, we might be tempted to use this directly in equations 2 and 1. But a number of problems must first be addressed:

• Equation 2 requires the **inverse** covariance matrix, whereas $C_{\mathcal{K}_i}$ is likely to be ill-conditioned. This situation will definitely arise if we use a *hard k-neighborhood* with $k < n$. In this case we get a Gaussian that is totally flat outside of the affine subspace spanned by $x_i$ and its $k$ neighbors, and it does not constitute a proper density in $\mathbb{R}^n$. A common way to deal with this problem is to add a small isotropic (spherical) Gaussian noise of variance $\sigma^2$ in all directions, which is done by simply adding $\sigma^2$ to the diagonal of the covariance matrix: $C_i = C_{\mathcal{K}_i} + \sigma^2 I$.

• Even if we regularize $C_i$ by adding $\sigma^2$, when we deal with high dimensional spaces, it would be prohibitive in computation time and storage to keep and use the full inverse covariance matrix as expressed in 2. This would in effect multiply both the time and storage requirement of the already expensive ordinary Parzen Windows by $n + 1$. So instead, we use a different, more compact representation of the inverse Gaussian, by storing only the eigenvectors associated with the first few largest eigenvalues of $C_i$, as described below.

The eigen-decomposition of a covariance matrix $C$ can be expressed as: $C = VDV'$, where the columns of $V$ are the orthonormal eigenvectors and $D$ is a diagonal matrix with the eigenvalues $\lambda_1 \ldots \lambda_n$, that we will suppose sorted in decreasing order, without loss of generality.

The first $d$ eigenvectors with largest eigenvalues correspond to the *principal directions* of the local neighborhood, i.e. the high variance local directions of the supposed underlying $d$-dimensional manifold (but the true underlying dimension is unknown and may actually vary across space). The last few eigenvalues and eigenvectors are but noise directions with a small variance. So we may, without too much risk, force those last few components to the same low noise level $\sigma^2$. We have done this by zeroing the last $n - d$ eigenvalues (by considering only the first $d$ leading eigenvalues) and then adding $\sigma^2$ to all eigenvalues. This allows us to store only the first $d$ eigenvectors, and to later compute $\mathcal{N}_{\mu, C}(x)$ in time $\mathcal{O}(n \cdot d)$ instead of $\mathcal{O}(n^2)$. Thus both the storage requirement and the computational cost when estimating the density at a test point is only about $d + 1$ times that of ordinary Parzen. It can easily be shown that such an approximation of the covariance matrix yields to the following computation of $\mathcal{N}_{\mu, C}(x)$:

---

**Algorithm LocalGaussian**$(x, x_i, V_i, \lambda_i, d, \sigma^2)$

**Input**: test vector $x \in \mathbb{R}^n$, training vector $x_i \in \mathbb{R}^n$, $d$ eigenvalues $\lambda_{ij}$, $d$ eigenvectors in the columns of $V_i$, dimension $d$, and the regularization hyper-parameter $\sigma^2$.

**(1)** $r = d \log(2\pi) + \sum_{j=1}^{d} \log(\lambda_j + \sigma^2) + (n - d) \log(\sigma^2)$

**(2)** $q = \frac{1}{\sigma^2} \|x - x_i\|^2 + \sum_{j=1}^{d} (\frac{1}{\lambda_j} - \frac{1}{\sigma^2}) \|V_j'(x - x_i)\|^2$

**Output**: Gaussian density $e^{-0.5(r+q)}$

---

In the case of the *hard k-neighborhood*, the training algorithm pre-computes the local principal directions $V_i$ of the $k$ nearest neighbors of each training point $i$ (in practice we compute them with a SVD rather than an eigen-decomposition of the covariance matrix, see below). Note that with $d = 0$, we trivially obtain the traditional Parzen windows estimator.

---

**Algorithm MParzen::Train**$(X, d, k, \sigma^2)$

**Input**: training set matrix $X$ with $l$ rows $x_i \in \mathbb{R}^n$, chosen number of principal directions $d$, chosen number of neighbors $k \geq d$, and regularization hyper-parameter $\sigma^2$.

**(1)** For $i \in \{1, 2, \ldots, l\}$
**(2)**      Collect $k$ nearest neighbors $x_j$ of $x_i$, and put $x_j - x_i$ in the rows of matrix $M$.
**(3)**      Perform a partial singular value decomposition of $M$, to obtain the leading $d$ singular values $s_j$ ($j \in \{1, \ldots, d\}$) and singular column vectors $V_{i \cdot j}$ of $M$.
**(4)**      For $j \in \{1, \ldots, d\}$, let $\lambda_{ij} = \sigma^2 + \frac{s_j^2}{l}$
**Output**: The model $\mathcal{M} = (X, V, \lambda, k, d, \sigma^2)$, where $V$ is an $l \times n \times d$ tensor that collects all the eigenvectors and $\lambda$ is a $l \times d$ matrix with all the eigenvalues.

---

**Algorithm MParzen::Test**$(x, \mathcal{M})$

**Input**: test point $x$ and model $\mathcal{M} = (X, V, \lambda, k, d, \sigma^2)$.

**(1)** $s \leftarrow 0$
**(2)** For $i \in \{1, 2, \ldots, l\}$
**(3)**      $s \leftarrow s + $ LocalGaussian$(x, x_i, V_i, \lambda_i, d, \sigma^2)$
**Output**: manifold Parzen estimator $\hat{p}_{mp}(x) = \frac{s}{l}$.

---

## 3   Related work

As we have already pointed out, Manifold Parzen Windows, like traditional Parzen Windows and so many other density estimation algorithms, results in defining the density as a mixture of Gaussians. What differs is mostly *how* those Gaussians and their parameters are chosen. The idea of having a parameterization of each Gaussian that orients it along the local principal directions also underlies the already mentioned work on mixtures of Gaussian pancakes [3], mixtures of probabilistic PCAs [4], and mixtures of factor analysers [5, 6]. All these algorithms typically model the density using a relatively small number of Gaussians, whose centers and parameters must be learnt with some iterative optimisation algorithm such as EM (procedures which are known to be sensitive to local minima traps). By contrast our approach is, like the original Parzen windows, heavily memory-based. It avoids the problem of optimizing the centers by assigning a Gaussian to every training point, and uses simple analytic SVD to compute the local principal directions for each.

Another successful memory-based approach that uses local directions and inspired our work is the tangent distance algorithm [7]. While this approach was initially aimed at solving classification tasks with a nearest neighbor paradigm, some work has already been done in developing it into a probabilistic interpretation for mixtures with a few gaussians,

as well as for full-fledged kernel density estimation [8, 9]. The main difference between our approach and the above is that the Manifold Parzen estimator does not require prior knowledge, as it infers the local directions directly from the data, although it should be easy to also incorporate prior knowledge if available.

We should also mention similarities between our approach and the *Local Linear Embedding* and recent related dimensionality reduction methods [10, 11, 12, 13]. There are also links with previous work on locally-defined *metrics* for nearest-neighbors [14, 15, 16, 17]. Lastly, it can also be seen as an extension along the line of traditional variable and adaptive kernel estimators that adapt the kernel width locally (see [18] for a survey).

## 4 Experimental results

Throughout this whole section, when we mention Parzen Windows (sometimes abbreviated *Parzen* ), we mean ordinary Parzen windows using a spherical Gaussian kernel with a single hyper-parameter $\sigma$, the width of the Gaussian.

When we mention Manifold Parzen Windows (sometimes abbreviated *MParzen*), we used a *hard k-neighborhood*, so that the hyper-parameters are: the number of neighbors $k$, the number of retained principal components $d$, and the additional isotropic Gaussian noise parameter $\sigma$.

When measuring the quality of a density estimator $\hat{p}(x)$, we used the average negative log likelihood: ANLL $= -\frac{1}{m} \sum_{i=1}^{m} \log \hat{p}(x_i)$ with the $m$ examples $x_i$ from a test set.

### 4.1 Experiment on 2D artificial data

A training set of 300 points, a validation set of 300 points and a test set of 10000 points were generated from the following distribution of two dimensional $(x, y)$ points:

$$x = 0.04 \, t \, \sin(t) + \epsilon_x, \ y = 0.04 \, t \, \cos(t) + \epsilon_y$$

where $t \sim U(3, 15)$, $\epsilon_x \sim N(0, 0.01)$, $\epsilon_y \sim N(0, 0.01)$, $U(a, b)$ is uniform in the interval $(a, b)$ and $N(\mu, \sigma)$ is a normal density.

We trained an ordinary Parzen, as well as MParzen with $d = 1$ and $d = 2$ on the training set, tuning the hyper-parameters to achieve best performance on the validation set. Figure 2 shows the training set and gives a good idea of the densities produced by both kinds of algorithms (as the visual representation for MParzen with $d = 1$ and $d = 2$ did not appear very different, we show only the case $d = 1$). The graphic reveals the anticipated "bumpyness" artifacts of ordinary Parzen, and shows that MParzen is indeed able to better concentrate the probability density along the manifold, even when the training data is scarce.

Quantitative comparative results of the two models are reported in table 1

**Table 1:** *Comparative results on the artificial data (standard errors are in parenthesis).*

| Algorithm | Parameters used | ANLL on test-set |
|---|---|---|
| Parzen | $\sigma = 0.0173$ | -1.183 (0.016) |
| MParzen | $d = 1, k = 11, \sigma = 0.09$ | -1.466 (0.009) |
| MParzen | $d = 2, k = 10, \sigma = 0.00001$ | -1.419 (0.009) |

Several points are worth noticing:

- Both *MParzen* models seem to achieve a lower ANLL than ordinary Parzen (even though the underlying manifold really has dimension $d = 1$), and with more consistency over the test sets (lower standard error).

- The optimal width $\sigma$ for ordinary *Parzen* is much larger than the noise parameter of the true generating model (0.01), probably because of the finite sample size.

- The optimal regularization parameter $\sigma$ for *MParzen* with $d = 1$ (i.e. supposing a one-dimensional underlying manifold) is very close to the actual noise parameter of the true generating model. This suggests that it was able to capture the underlying structure quite well. Also it is the best of the three models, which is not surprising, since the true model is indeed a one dimensional manifold with an added isotropic Gaussian noise.

- The optimal additional noise parameter $\sigma$ for *MParzen* with $d = 2$ (i.e. supposing a two-dimensional underlying manifold) is close to 0, which suggests that the model was able to capture all the noise in the second "principal direction".

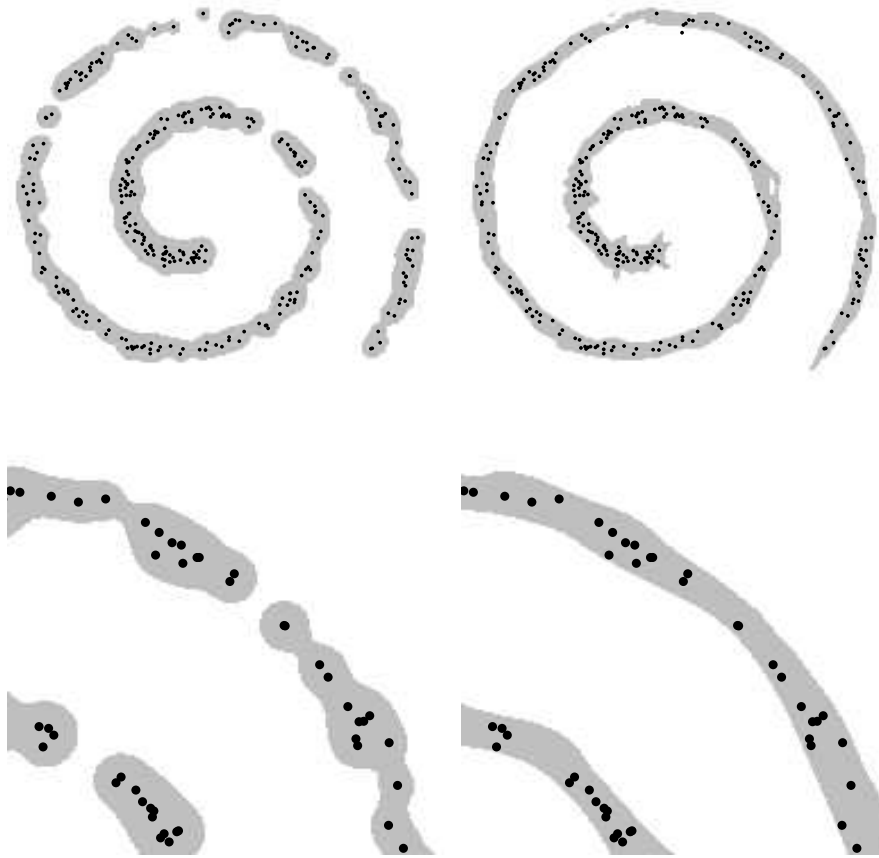

**Figure 2:** *Illustration of the density estimated by ordinary Parzen Windows (left) and Manifold Parzen Windows (right). The two images on the bottom are a zoomed area of the corresponding image at the top. The 300 training points are represented as black dots and the area where the estimated density $\hat{p}_{mp}(x)$ is above 1.0 is painted in gray. The excessive "bumpyness" and holes produced by ordinary Parzen windows model can clearly be seen, whereas Manifold Parzen density is better aligned with the underlying manifold, allowing it to even successfully "extrapolate" in regions with few data points but high true density.*

### 4.2 Density estimation on OCR data

In order to compare the performance of both algorithms for density estimation on a real-world problem, we estimated the density of one class of the MNIST OCR data set, namely the "2" digit. The available data for this class was divided into 5400 training points, 558 validation points and 1032 test points. Hyper-parameters were tuned on the validation set. The results are summarized in Table 2, using the performance measures introduced above (average negative log-likelihood). Note that the improvement with respect to Parzen windows is extremely large and of course statistically significant.

**Table 2:** *Density estimation of class '2' in the MNIST data set. Standard errors in parenthesis.*

| Algorithm | Parameters used | validation ANLL | test ANLL |
|---|---|---|---|
| Parzen | $\sigma = 0.19$ | -197.27 (4.18) | -197.19 (3.55) |
| MParzen | $d = 50, k = 80, \sigma = 0.09$ | -696.42 (5.94) | -695.15 (5.21) |

### 4.3 Classification performance

To obtain a probabilistic classifier with a density estimator we train an estimator $\hat{p}_c(x) = \hat{p}(x|c)$ for each class $c$, and apply Bayes' rule to obtain $\hat{P}(c|x) = \frac{\hat{p}(x|c)\hat{P}(c)}{\sum_{c'} \hat{p}(x|c')\hat{P}(c')}$. When measuring the quality of a probabilistic classifier $\hat{P}(c|x)$, we used the negative conditional log likelihood: ANCLL $\stackrel{\text{def}}{=} -\frac{1}{m} \sum_{i=1}^{m} \log \hat{P}(c_i|x_i)$, with the $m$ examples $(c_i, x_i)$ (correct class, input) from a test set.

This method was applied to both the Parzen and the Manifold Parzen density estimators, which were compared with state-of-the-art Gaussian SVMs on the full USPS data set. The original training set (7291) was split into a training (first 6291) and validation set (last 1000), used to tune hyper-parameters. The classification errors for all three methods are compared in Table 3, where the hyper-parameters are chosen based on validation classification error. The log-likelihoods are compared in Table 4, where the hyper-parameters are chosen based on validation ANCLL. Hyper-parameters for SVMs are the box constraint $C$ and the Gaussian width $\sigma$. MParzen has the lowest classification error and ANCLL of the three algorithms.

**Table 3:** *Classification error obtained on USPS with SVM, Parzen windows and Manifold Parzen windows classifiers.*

| Algorithm | validation error | test error | parameters |
|---|---|---|---|
| SVM | 1.2% | 4.68% | $C = 100, \sigma = 8$ |
| Parzen | 1.8% | 5.08% | $\sigma = 0.8$ |
| MParzen | 0.9% | 4.08% | $d = 11, k = 11, \sigma^2 = 0.1$ |

**Table 4:** *Comparative negative conditional log likelihood obtained on USPS.*

| Algorithm | valid ANCLL | test ANCLL | parameters |
|---|---|---|---|
| Parzen | 0.1022 | 0.3478 | $\sigma = 0.8$ |
| MParzen | 0.0658 | 0.3384 | $d = 17, k = 17, \sigma^2 = 0.75$ |

## 5 Conclusion

The rapid increase in computational power now allows to experiment with sophisticated non-parametric models such as those presented here. They have allowed to show the usefulness of learning the local structure of the data through a regularized covariance matrix estimated for each data point. By taking advantage of local structure, the new kernel density estimation method outperforms the Parzen windows estimator. Classifiers built from

this density estimator yield state-of-the-art knowledge-free performance, which is remarkable for a not discriminatively trained classifier. Besides, in some applications, the accurate estimation of probabilities can be crucial, e.g. when the classes are highly imbalanced.

Future work should consider other alternative methods of estimating the local covariance matrix, for example as suggested here using a weighted estimator, or taking advantage of prior knowledge (e.g. the Tangent distance directions).

# References

[1] P. Vincent and Y. Bengio. K-local hyperplane and convex distance nearest neighbor algorithms. In T.G. Dietterich, S. Becker, and Z. Ghahramani, editors, *Advances in Neural Information Processing Systems*, volume 14. The MIT Press, 2002.

[2] E. Parzen. On the estimation of a probability density function and mode. *Annals of Mathematical Statistics*, 33:1064–1076, 1962.

[3] G.E. Hinton, M. Revow, and P. Dayan. Recognizing handwritten digits using mixtures of linear models. In G. Tesauro, D.S. Touretzky, and T.K. Leen, editors, *Advances in Neural Information Processing Systems 7*, pages 1015–1022. MIT Press, Cambridge, MA, 1995.

[4] M.E. Tipping and C.M. Bishop. Mixtures of probabilistic principal component analysers. *Neural Computation*, 11(2):443–482, 1999.

[5] Z. Ghahramani and G.E. Hinton. The EM algorithm for mixtures of factor analyzers. Technical Report CRG-TR-96-1, Dpt. of Comp. Sci., Univ. of Toronto, 21 1996.

[6] Z. Ghahramani and M. J. Beal. Variational inference for Bayesian mixtures of factor analysers. In *Advances in Neural Information Processing Systems 12*, Cambridge, MA, 2000. MIT Press.

[7] P. Y. Simard, Y. A. LeCun, J. S. Denker, and B. Victorri. Transformation invariance in pattern recognition — tangent distance and tangent propagation. *Lecture Notes in Computer Science*, 1524, 1998.

[8] D. Keysers, J. Dahmen, and H. Ney. A probabilistic view on tangent distance. In *22nd Symposium of the German Association for Pattern Recognition*, Kiel, Germany, 2000.

[9] J. Dahmen, D. Keysers, M. Pitz, and H. Ney. Structured covariance matrices for statistical image object recognition. In *22nd Symposium of the German Association for Pattern Recognition*, Kiel, Germany, 2000.

[10] S. Roweis and L. Saul. Nonlinear dimensionality reduction by locally linear embedding. *Science*, 290(5500):2323–2326, Dec. 2000.

[11] Y. Whye Teh and S. Roweis. Automatic alignment of local representations. In S. Becker, S. Thrun, and K. Obermayer, editors, *Advances in Neural Information Processing Systems*, volume 15. The MIT Press, 2003.

[12] V. de Silva and J.B. Tenenbaum. Global versus local approaches to nonlinear dimensionality reduction. In S. Becker, S. Thrun, and K. Obermayer, editors, *Advances in Neural Information Processing Systems*, volume 15. The MIT Press, 2003.

[13] M. Brand. Charting a manifold. In S. Becker, S. Thrun, and K. Obermayer, editors, *Advances in Neural Information Processing Systems*, volume 15. The MIT Press, 2003.

[14] R. D. Short and K. Fukunaga. The optimal distance measure for nearest neighbor classification. *IEEE Transactions on Information Theory*, 27:622–627, 1981.

[15] J. Myles and D. Hand. The multi-class measure problem in nearest neighbour discrimination rules. *Pattern Recognition*, 23:1291–1297, 1990.

[16] J. Friedman. Flexible metric nearest neighbor classification. Technical Report 113, Stanford University Statistics Department, 1994.

[17] T. Hastie and R. Tibshirani. Discriminant adaptive nearest neighbor classification and regression. In David S. Touretzky, Michael C. Mozer, and Michael E. Hasselmo, editors, *Advances in Neural Information Processing Systems*, volume 8, pages 409–415. The MIT Press, 1996.

[18] A.J. Inzenman. Recent developments in nonparametric density estimation. *Journal of the American Statistical Association*, 86(413):205–224, 1991.
